# Stimulus Evoked Independent Factor Analysis of MEG Data with Large Background Activity

**S.S. Nagarajan**
Biomagnetic Imaging Laboratory
Department of Radiology
University of California, San Francisco
San Francisco, CA 94122
sri@radiology.ucsf.edu

**H.T. Attias**
Golden Metallic, Inc.
P.O. Box 475608
San Francisco, CA 94147
htattias@goldenmetallic.com

**K.E. Hild**
Biomagnetic Imaging Laboratory
Department of Radiology
University of California, San Francisco
San Francisco, CA 94122
hild@mrsc.ucsf.edu

**K. Sekihara**
Dept. of Systems Design and Engineering
Tokyo Metropolitan University
Asahigaoka 6-6, Hino, Tokyo 191-0065
ksekiha@cc.tmit.ac.jp

## Abstract

This paper presents a novel technique for analyzing electromagnetic imaging data obtained using the stimulus evoked experimental paradigm. The technique is based on a probabilistic graphical model, which describes the data in terms of underlying evoked and interference sources, and explicitly models the stimulus evoked paradigm. A variational Bayesian EM algorithm infers the model from data, suppresses interference sources, and reconstructs the activity of separated individual brain sources. The new algorithm outperforms existing techniques on two real datasets, as well as on simulated data.

## 1 Introduction

Electromagnetic source imaging, the reconstruction of the spatiotemporal activation of brain sources from MEG and EEG data, is currently being used in numerous studies of human cognition, both in normal and in various clinical populations [1]. A major advantage of MEG/EEG over other noninvasive functional brain imaging techniques, such as fMRI, is the ability to obtain valuable information about neural dynamics with high temporal resolution on the order of milliseconds. An experimental paradigm that is very popular in imaging studies is the stimulus evoked paradigm. In this paradigm, a stimulus, e.g., a tone at a particular frequency and duration, is presented to the subject at a series of equally spaced time points. Each presentation (or trial) produces activity in a set of brain sources, which generates an electromagnetic field captured by the sensor array. These data constitute the stimulus evoked response, and analyzing them can help to gain insights into the mechanism used by the brain to process the stimulus and similar sensory inputs. This paper presents a new technique for analyzing stimulus evoked electromagnetic imaging data.

An important problem in analyzing such data is that MEG/EEG signals, which are captured by sensors located outside the brain, contain not only signals generated by brain sources evoked by the stimulus, but also interference signals, generated by other sources such as spontaneous brain activity, eye blinks and other biological and non-biological sources of artifacts. Interference signals overlap spatially and temporally with the stimulus evoked signals, making it difficult to obtain accurate reconstructions of evoked brain sources. A related problem is that signals from different evoked sources themselves overlap with each other, making it difficult to localize individual sources and reconstruct their separate responses.

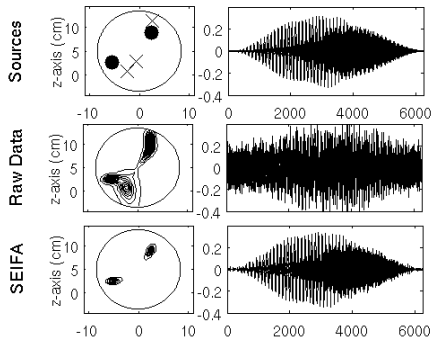

Figure 1: Simulation example (see text)

Many approaches have been taken to the problem of suppressing interference signals. One method is averaging over multiple trials, which reduces the contributions from interference sources, assuming that they are uncorrelated with the stimulus and that their autocorrelation time scale is shorter than the trial length. However, a successful application of this method requires a large number of trials, effectively limiting the number of stimulus conditions per experiment. It usually also requires manual rejection of trials containing conspicuous artifacts. A set of methods termed subspace techniques computes a projection of the sensor data onto the signal subspace, which corresponds to brain sources of interest. However, these methods rely on thresholding to determine the noise level, and tend to discard information below threshold. Consequently, those methods perform well only when the interference level is low.

Independent component analysis (ICA) techniques [4-8], introduced more recently, attempt to decompose the sensor data into a set of signals that are mutually statistically independent. Artifacts such as eye blinks are independent of brain source activity and ICA has been able in many cases to successfully separate the two types of signals into distinct groups of output variables. However, ICA techniques have several shortcomings. First, they require pre-processing the sensor data to reduce dimensionality from, which causes loss of information on brain sources with relatively low amplitude. This is because, for $K$ sensors, ICA must learn a square $K \times K$ unmixing matrix from $N$ data points; typical values such as $K = 275, N = 700$ can lead to poor performance due to local maxima, overfitting, and slow convergence. Second, ICA assumes $L + M = K'$, where $L, M$ are the number of evoked and interference sources and $K' < K$ is the *reduced* input dimensionality. However, many cases have $L + M > K'$, which leads to suboptimal and sometime failed separation. Third, ICA requires post-processing of its output signals, usually via manual examination by experts (though sometime by thresholding), to determine which signals correspond evoked brain sources of interest.

The fourth drawback of ICA techniques is that, by design, they cannot exploit the advantage offered by the evoked stimulus paradigm. Whereas interference sources are continuously active, evoked sources become active at each trial only near the time of stimulus presentation, termed stimulus onset time. Hence, knowledge of the onset times can help separate the evoked sources. However, the onset times, which are determined by the experimental design and available during data analysis, are ignored by ICA.

In this paper we present a novel technique for suppressing interference signals and separating signals from individual evoked sources. The technique is based on a new probabilistic graphical model termed *stimulus evoked independent factor analysis* (SEIFA). This model,

an extension of [2], describes the observed sensor data in terms of two sets of independent variables, termed factors, which are not directly observable. The factors in the first set represent evoked sources, and the factors in the second set represent interference sources. The sensor data are generated by linearly combining the factors in the two sets using two mixing matrices, followed by adding sensor noise. The mixing matrices and the precision matrix of the sensor noise constitute the SEIFA model parameters, and are inferred from data using a variational Bayesian EM algorithm [3], which computes their posterior distribution. Separation of the evoked sources is achieved in the course of processing by the algorithm.

The SEIFA model is free from the above four shortcomings. It can be applied directly to the sensor data without dimensionality reduction, therefore no information is lost. Rather than learning a square $K \times K$ unmixing matrix, it learns a $K \times (L + M)$ mixing matrix, where the number of interference factors $M$ is minimized using automatic Bayesian model selection which is part of the algorithm. In addition, SEIFA is designed to explicitly model the stimulus evoked paradigm, hence it optimally exploits the knowledge of stimulus onset times. Consequently, evoked sources are automatically identified and no post-processing is required.

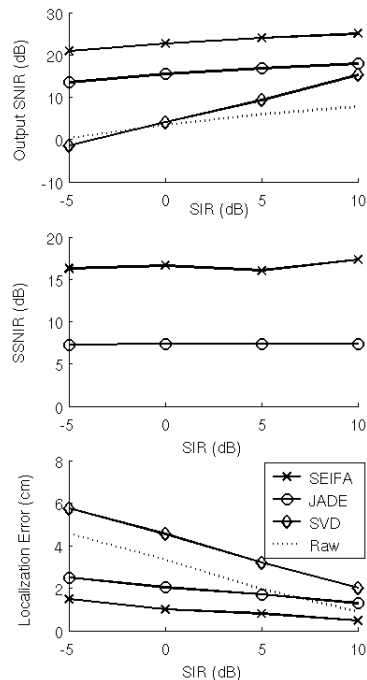

Figure 2: Performance on simulated data (see text)

## 2   SEIFA
## Probabilistic Graphical Model

This section presents the SEIFA probabilistic graphical model, which is the focus of this paper. The SEIFA model describes observed MEG sensor data in terms of three types of underlying, unobserved signals: (1) signals arising from stimulus evoked sources, (2) signals arising from interference sources, and (2) sensor noise signals. The model is inferred from data by an algorithm presented in the next section. Following inference, the model is used to separate the evoked source signals from those of the interference sources and from sensor noise, thus providing a clean version of the evoked response. The model further separates the evoked response into statistically independent factors. In addition, it produces a regularized correlation matrix of the clean evoked response and of each independent factors, which facilitates localization.

Let $y_{in}$ denote the signal recorded by sensor $i = 1 : K$ at time $n = 1 : N$. We assume that these signals arise from $L$ evoked factors and $M$ interference factors that are combined linearly. Let $x_{jn}$ denote the signal of evoked factor $j = 1 : L$, and let $u_{jn}$ denote the signal of interference factor $j = 1 : M$, both at time $n$. We use the term factor rather than source for a reason explained below. Let $A_{ij}$ denote the evoked mixing matrix, and let $B_{ij}$ denote the interference mixing matrix. Those matrices contain the coefficients of the linear combination of the factors that produces the data. They are analogous to the factor loading matrix in the factor analysis model. Let $v_{in}$ denote the noise signal on sensor $i$.

We use an evoked stimulus paradigm, where a stimulus is presented at a specific time, termed the stimulus onset time, and is absent beforehand. The stimulus onset time is de-

fined as $n = N_0 + 1$. The period preceding the onset $n = 1 : N_0$ is termed pre-stimulus period, and the period following the onset $n = N_0 + 1 : N$ is termed post-stimulus period. We assume the evoked factors are active only post stimulus and satisfy $x_{jn} = 0$ before its onset. Hence

$$y_n = \begin{cases} Bu_n + v_n, & n = 1 : N_0 \\ Ax_n + Bu_n + v_n, & n = N_0 + 1 : N \end{cases} \tag{1}$$

To turn (1) into a probabilistic model, each signal must be modelled by a probability distribution. Here, each evoked factor is modelled by a mixture of Gaussian (MOG) distributions. For factor $j$ we have a MOG model with $S_j$ components, also termed states,

$$p(x_n) = \prod_{j=1}^{L} p(x_{jn}) , \qquad p(x_{jn}) = \sum_{s_j=1}^{S_j} \mathcal{N}(x_{jn} \mid \mu_{j,s_j}, \nu_{j,s_j}) \pi_{j,s_j} \tag{2}$$

State $s_j$ is a Gaussian with mean $\mu_{j,s_j}$ and precision $\nu_{j,s_j}$, and its probability is $\pi_{j,s_j}$. We model the factors as mutually statistically independent.

There are three reasons for using MOG distributions, rather than Gaussians, to describe the evoked factors. First, evoked brain sources are often characterized by spikes or by modulated harmonic functions, leading to non-Gaussian distributions. Second, previous work on ICA has shown that independent Gaussian sources that are linearly mixed cannot be separated. Since we aim to separate the evoked response into contributions from individual factors, we must therefore use independent non-Gaussian factor distributions. Third, as is well known, a MOG model with a suitably chosen number of states can describe arbitrary distributions at the desired level of accuracy.

For interference signals and sensor noise we employ a Gaussian model. Each interference factor is modelled by an independent, zero-mean Gaussian distribution with unit precision,

$$p(u_n) = \prod_{j=1}^{M} \mathcal{N}(u_{jn} \mid 0, 1) = \mathcal{N}(u_n \mid 0, I) \tag{3}$$

The Gaussian model implies that we exploit only second order statistics of the interference signals. This contrasts with the evoked signals, whose MOG model facilitates exploiting higher order statistics, leading to more accurate reconstruction and to separation.

The sensor noise is modelled by a zero-mean Gaussian distribution with a diagonal precision matrix $\lambda$, $p(v_n) = \mathcal{N}(v_n \mid 0, \lambda)$. From (1) we obtain $p(y_n \mid x_n, u_n) = p(v_n)$ where we substitute $v_n = y_n - Ax_n - Bu_n$ with $x_n = 0$ for $n = 1 : N_0$. Hence, we obtain the distribution of the sensor signals conditioned on the evoked and interference factors,

$$p(y_n \mid x_n, u_n, A, B) = \begin{cases} \mathcal{N}(y_n \mid Bu_n, \lambda), & n = 1 : N_0 \\ \mathcal{N}(y_n \mid Ax_n + Bu_n, \lambda), & n = N_0 + 1 : N \end{cases} \tag{4}$$

SEIFA also makes an i.i.d. assumption, meaning the signals at different time points are independent. Hence $p(y, x, u \mid A, B) = \prod_n p(y_n \mid x_n, u_n, A, B) p(x_n) p(u_n)$. where $y, x, u$ denote collectively the signals $y_n, x_n, u_n$ at all time points. The i.i.d. assumption is made for simplicity, and implies that the algorithm presented below can exploit the spatial statistics of the data but not their temporal statistics.

To complete the definition of SEIFA, we must specify prior distributions over the model parameters. For the noise precision matrix $\lambda$ we choose a flat prior, $p(\lambda) = const$. For the mixing matrices $A, B$ we choose to use a conjugate prior

$$p(A) = \prod_{ij} \mathcal{N}(A_{ij} \mid 0, \lambda_i \alpha_j) , \qquad p(B) = \prod_{ij} \mathcal{N}(B_{ij} \mid 0, \lambda_i \beta_j) \tag{5}$$

where all matrix elements are independent zero-mean Gaussians and the precision of the $ij$th matrix element is proportional to the noise precision $\lambda_i$ on sensor $i$. It is the $\lambda$ dependence which makes this prior conjugate. The proportionality constants $\alpha_j$ and $\beta_j$ constitute the parameters of the prior, a.k.a. hyperparameters. Eqs. (2,3,4,5) fully define the SEIFA model.

## 3 Inferring the SEIFA Model from Data: A VB-EM Algorithm

This section presents an algorithm that infers the SEIFA model from data. SEIFA is a probabilistic model with hidden variables, since the evoked and interference factors are not directly observable, hence it must be treated in the EM framework. We use variational Bayesian EM (VB-EM), which has two relevant advantages over standard EM. First, it is more robust to overfitting, which can be a significant problem when working with high-dimensional but relatively short time series (here we analyze $N < 1000$ point long, $K = 275$ dimensional data sequences). To achieve this robustness, VB-EM computes (using a variational approximation) a full posterior distribution over model parameters, rather than a single MAP estimate. This means that VB-EM considers all possible parameters values, and computes the probability of each value conditioned on the observed data. It also performs automatic model order selection by optimizing the hyperparameters, and consequently uses the minimum number of parameters needed to explain the data. Second, VB-EM produces automatically regularized estimators for the evoked response correlation matrices (required for source localization), where standard EM produces poorly conditioned ones. This is also a result of computing a parameter posterior.

VB-EM is an iterative algorithm, where each iteration consists of an E- and an M-step.

**E-step**. For the pre-stimulus period $n = 1 : N_0$ we compute the posterior over the interference factors $u_n$ only. It is a Gaussian distribution with posterior mean $\bar{u}_n$ and covariance $\Phi$ given by

$$\bar{u}_n = \Phi \bar{B}^T \lambda y_n \ , \quad \Phi = \left( \bar{B}^T \lambda \bar{B} + I + K\Psi_{BB} \right)^{-1} \tag{6}$$

where $\bar{B}$ are $\Psi_{BB}$ are the posterior mean and covariance of the interference mixing matrix $B$ computed in the M-step below (more precisely, the posterior covariance of the $i$th row of $B$ is $\Psi_{BB}/\lambda_i$).

For the post-stimulus period $n = N_0 + 1 : N$ we compute the posterior over the evoked and interference factors $x_n, u_n$, and the collective state $s_n$ of the evoked factors. The latter is defined by the $L$-dimensional vector $s_n = (s_{1n}, s_{2n}, ..., s_{Ln})$, where $s_{jn} = 1 : S_j$ is the state of evoked factor $j$ at time $n$. The total number of collective states is $S = \prod_j S_j$.

To simplify the notation, we combine the evoked and interference factors into a single $L' \times 1$ vector $x'_n = (x_n, u_n)$, where $L' = L + M$, and their mixing matrices into a single $K \times L'$ matrix $A' = (A, B)$. Now, at time $n$, let $r$ run over all the $S$ collective states. For each $r$, the posterior over the factors conditioned on $s_n = r$ is Gaussian, with posterior mean $\bar{x}_{rn}, \bar{u}_{rn}$ and covariance $\Gamma_r$ given by

$$\bar{x}'_{rn} = \Gamma_r \left( \bar{A}'^T \lambda y_n + \nu'_r \mu'_r \right) \ , \quad \Gamma_r = \left( \bar{A}'^T \lambda \bar{A}' + \nu'_r + K\Psi \right)^{-1} \tag{7}$$

We have defined $\bar{x}'_{rn} = (\bar{x}_{rn}, \bar{u}_{rn})$ and $\bar{A}' = (\bar{A}, \bar{B})$. The $L \times 1$ vector $\mu'_r$ and the diagonal $L \times L$ matrix $\nu'_r$ contain the means and precisions of the individual states (see (2)) composing $r$. The posterior mean and covariance $\bar{A}', \Psi$ are computed in the M-step. Next, compute the posterior probability that $s_n = r$ by

$$\bar{\pi}_{rn} = \frac{1}{z_n} \pi_r \sqrt{\mid \nu_r \mid\mid \Gamma_r \mid} \exp \left( -\frac{1}{2} y_n^T \lambda y_n + \frac{1}{2} \mu_r^T \nu_r \mu_r - \frac{1}{2} \bar{x}'_{rn} \Gamma_r^{-1} \bar{x}'_{rn} \right) \tag{8}$$

where $z_n$ is a normalization constant and $\mu_r, \nu_r, \pi_r$ are the MOG parameters of (2).

**M-step.** We divide the model parameters into two sets. The first set includes the mixing matrices $A, B$, for which we compute full posterior distributions. The second set includes the noise precision $\lambda$ and the diagonal hyperparameters matrices $\alpha, \beta$, for which we compute MAP estimates. The posterior over $A, B$ is Gaussian factorized over their rows, where the mean is

$$\begin{aligned} \bar{A} &= R_{yx}\Psi \\ \bar{B} &= R_{yu}\Psi \end{aligned} \quad , \qquad \Psi = \left( \begin{array}{cc} R_{xx} + \alpha & R_{xu} \\ R_{xu}^T & R_{uu} + \beta \end{array} \right)^{-1} \tag{9}$$

and where the $i$th row of $A' = (A, B)$ has covariance $\Psi/\lambda_i$. The hyperparameters $\alpha_j, \beta_j$ are diagonal entries of diagonal matrices $\alpha, \beta$. $R_{yx}, R_{yu}, R_{xx}, R_{xu}, R_{uu}$ are posterior correlations between the factors and the data and among the factors themselves, e.g., $R_{yx} = \sum_n \langle y_n x_n \rangle, R_{xx} = \sum_n \langle x_n x_n \rangle$, where $\langle \cdot \rangle$ denotes posterior averaging. They are easily computed in terms of the E-step quantities $\bar{u}_n, \bar{x}'_{rn}, \Phi, \Gamma_r, \bar{\pi}_{rn}$ and are omitted.

Next, the hyperparameter matrices $\alpha, \beta$ are updated by

$$\alpha^{-1} = \mathrm{diag}\left( \bar{A}^T \lambda \bar{A}/K + \Psi_{AA} \right) \ , \qquad \beta^{-1} = \mathrm{diag}\left( \bar{B}^T \lambda \bar{B}/K + \Psi_{BB} \right) \tag{10}$$

and the noise precision matrix by $\lambda^{-1} = \mathrm{diag}(R_{yy} - \bar{A}R_{yx}^T - \bar{B}R_{yu}^T)/N$. $\Psi_{AA}$ and $\Psi_{BB}$ are the appropriate blocks of $\Psi$ in (9). The interference mixing matrix and the noise precision are initialized from pre-stimulus data. We used MOG parameters corresponding to peaky (super-Gaussian) distributions.

**Estimating and Localizing Clean Evoked Responses.** Let $z_{in}^j = \langle A_{ij}x_{jn} \rangle$ denote the inferred individual contribution from evoked factor $j$ to sensor signal $i$. It is given via posterior averaging by

$$\bar{z}_{in}^j = \bar{A}_{ij}\bar{x}_{jn} \tag{11}$$

where $\bar{x}_n = \sum_r \bar{\pi}_r \bar{x}_{rn}$. Computing this estimate amounts to obtaining a clean version of the individual contribution from each factor and of their combined contribution, and removing contributions from interference factors and sensor noise.

The localization of individual evoked factors using sensor signals $z_n^j$ can be achieved by many algorithms. In this paper, we use adaptive spatial filters that take data correlation matrices as inputs for localization, because these methods have been shown to have superior spatial resolution and non-zero localization bias [6]. Let $C^j = \sum_n \langle z_n^j (z_n^j)^T \rangle$ denote the inferred sensor data correlation matrix corresponding to the individual contribution from evoked factor $j$. Then,

$$C^j = \left[ \bar{A}^j (\bar{A}^j)^T + \lambda^{-1}(\Psi_{AA})_{jj} \right] (R_{xx})_{jj} \tag{12}$$

where $\bar{A}^j$ is a $K \times 1$ vector denoting the $j$th column of $\bar{A}$. Notice that the VB-EM approach has produced a correlation matrix that is automatically regularized (due to the $\Psi_{AA}$ term) and can be used for localization in its current form. In contrast, computing it from the signal estimates obtained by other methods, such as PCA or ICA, yields a poorly conditioned matrix that requires post-processing.

## 4  Experiments on Real and Simulated Data

**Simulations.** Fig. 1 shows a simulation with two evoked sources and three interference sources with $N = 10000$, signal-to-interference (SIR) of 0 dB and signal-to-sensor-noise (SNR) of 5dB. The true locations of the evoked sources, each of which is denoted by $\bullet$, and the true locations of the background sources, each of which is denoted by $\times$ are shown in the top left panel. The right column in the top row shows the time courses of the evoked sources as they appear at the sensors. The time courses of the actual sensor signals, which also include the effects of background sources and sensor noise, are shown in the middle row (right column). The bottom row shows the localization and time-course of cleaned

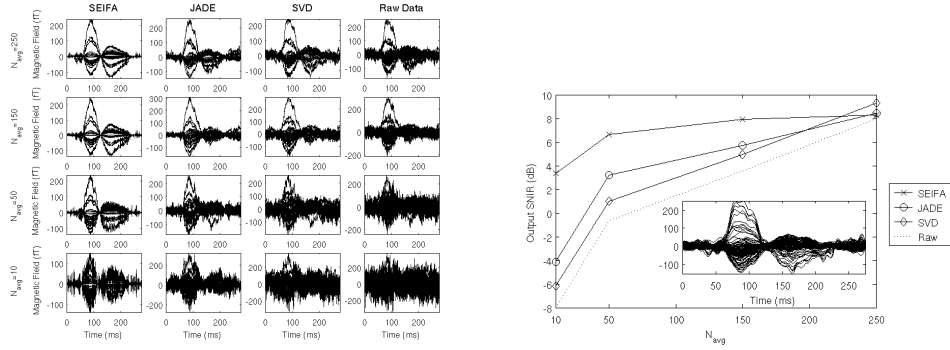

Figure 3: Estimating auditory-evoked responses from small trial averages (see text)

evoked sources estimated using SEIFA, which agrees with the true location and time-course. Fig. 2 shows the mean performance as a function of SIR, across 50 Monte Carlo trials for $N = 1000$ and SNR of 10 dB, for different locations of evoked and interference sources. Denoising performance is quantified by the output signal-to-(noise+interference) ratio (SNIR) and shown in the top panel. SEIFA outperforms both our benchmark methods, providing a 5-10 dB improvement over JADE [7] and SVD. Separation performance of individual evoked factors is quantified by (separated-signal)-to-(noise+interference) ratio (SSNIR) (definition omitted) and is shown in the middle panel. SEIFA far outperforms JADE for this set of examples. JADE is able to separate the background sources from the evoked sources (hence gives good denoising performance), but it is not always able to separate the evoked sources from each other. The Infomax algorithm [4] (results not shown) exhibited poor separation performance similar to JADE. Finally, localization performance is quantified by the mean distance in cm between the true evoked source locations and the estimated locations, as shown in the bottom panel. Here too, SEIFA far outperforms all other methods, especially for low SIR. Notably, SEIFA performance appears to be quite robust to the i.i.d. assumption of the evoked and background sources, because in these simulations evoked sources were assumed to be damped sinusoids and interference sources were sinusoids.

## 4.1 Real Data

**Denoising averages from small number of trials.** Auditory evoked responses from a particular subject obtained by averaging different number of trials are shown in figure 3 (left panel). SEIFA is able to clearly recover responses even from small trial averages. To quantify the performance of the different methods, a filtered version of the raw data for $N_{avg} = 250$ was assumed as "ground-truth", and is shown in the inset of the right panel. The output SNIR as a function of $N_{avg}$ is also shown in figure 3 (right panel). SEIFA exhibits the best performance especially for small trial averages.

**Separation of evoked sources.** To highlight SEIFA's ability to separately localize evoked sources, we conducted an experiment involving simultaneous presentation of auditory and somatosensory stimuli. We expected the activation of contralateral auditory and somatosensory cortices to overlap in time. A pure tone (400ms duration, 1kHz, 5 ms ramp up/down) was presented binaurally with a delay of 50 ms following a pneumatic tap on the left index finger. Averaging is performed over $N_{avg} = 100$ trials triggered on the onset of the tap. Results from SEIFA for this experiment are shown in Figure 4. In these figures, one panel shows a contour map that shows the polarity and magnitude of the denoised and raw sensor signals in sensor space. The contour plot of the magnetic field on the sensor array, corresponding to the mapping of three-dimensional sensor surface array to points within a

circle, shows the magnetic field profile at a particular instant of time relative to the stimulus presentation. Other panels show localization of a particular evoked factor overlaid on the subjects' MRI. Three orthogonal projections - axial, sagittal and coronal MRI slices, that highlight all voxels having activity that is $> 80\%$ of maximum are shown. Results are based on the right hemisphere channels above contralateral somatosensory and auditory cortices. Localization of time-course of the first two factors estimated by SEIFA are shown in left and middle panels of figure 4. The first two factors localize to primary somatosensory cortex (SI), however with differential latencies. The first factor shows a peak response at a latency of 50 ms, whereas the second factor shows the response at a later latency. Interestingly, the third factor localizes to auditory cortex and the extracted time-course corresponds well to an auditory evoked response that is well-separated from the somatosensory response (figure 3 right panels).

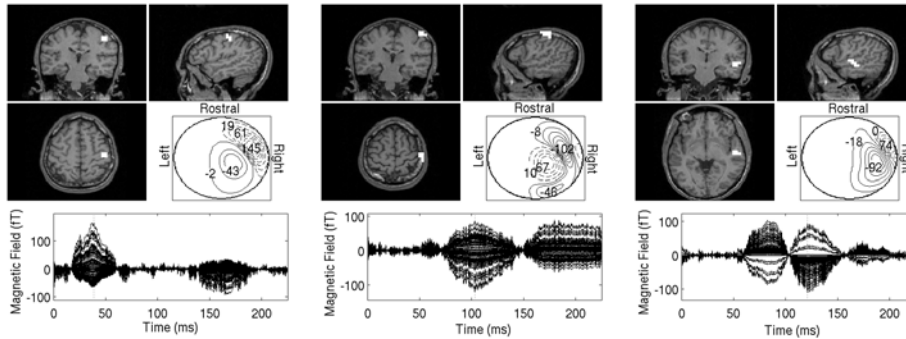

Figure 4: Estimated SEIFA factors for auditory-somatosensory experiment

## 5 Extensions

Whereas this paper uses fixed values for the number of evoked and interference sources $L, M$ (though the effective number of interference sources was determined via optimizing the hyperparameter $\beta$), VB-EM facilitates inferring them from data, and we plan to investigate the effectiveness of this procedure. We also plan to infer the distribution of evoked sources (MOG parameters) from data rather than using a fixed distribution. Another extension that could enhance performance is exploiting temporal correlation in the data. We plan to do it by incorporating temporal (e.g., autoregressive) models into the source distributions and infer their parameters from data.

**References**

[1] S. Baillet, J. C. Mosher, and R. M. Leahy. Electromagnetic brain mapping.*Signal Processing Magazine*, 18:14-30, 2001.

[2] H. Attias (1999). Independent Factor Analysis. *Neur. Comp. 11*, 803-851.

[3] H. Attias (2000). A variational Bayes framework for graphical models. *Adv. Neur. Info. Proc. Sys. 12*, 209-215.

[4] T.-P. Jung, S. Makeig, M. Westerfield, J. Townsend, E. Courchesne, T.J. Sejnowski (2000). Removal of eye artifacts from visual event related potentials in normal and clinical subjects. *J. Clin. Neurophys. 40*, 516-520.

[5] S. Makeig, S. Debener, J. Onton, A. Delorme (2004). Mining event related brain dynamics. *Trends Cog. Sci. 8*, 204-210.

[6] K. Sekihara, S. Nagarajan, D. Poeppel, A. Marantz, Y. Miyashita (2001). Reconstructing spatio-temporal activities of neural sources using a MEG vector beamformer technique. *IEEE Trans. Biomed. Eng. 48*, 760-771.

[7] J.F.Cardoso (1999) High-order contrasts for independent component analysis, *Neural Computation*, 11(1):157–192.

[8] R. Vigario, J. Sarela, V. Jousmaki, M. Hamalainen, E. Oja (2000). Independent component approach to the analysis of EEG and MEG recordings. *IEEE Trans. Biomed. Eng. 47*, 589-593.
